# Learning visual motion in recurrent neural networks

**Marius Pachitariu, Maneesh Sahani**
Gatsby Computational Neuroscience Unit
University College London, UK
`{marius, maneesh}@gatsby.ucl.ac.uk`

## Abstract

We present a dynamic nonlinear generative model for visual motion based on a latent representation of binary-gated Gaussian variables. Trained on sequences of images, the model learns to represent different movement directions in different variables. We use an online approximate inference scheme that can be mapped to the dynamics of networks of neurons. Probed with drifting grating stimuli and moving bars of light, neurons in the model show patterns of responses analogous to those of direction-selective simple cells in primary visual cortex. Most model neurons also show speed tuning and respond equally well to a range of motion directions and speeds aligned to the constraint line of their respective preferred speed. We show how these computations are enabled by a specific pattern of recurrent connections learned by the model.

## 1 Introduction

Perhaps the most striking property of biological visual systems is their ability to efficiently cope with the high bandwidth data streams received from the eyes. Continuous sequences of images represent complex trajectories through the high-dimensional nonlinear space of two dimensional images. The survival of animal species depends on their ability to represent these trajectories efficiently and to distinguish visual motion on a fast time scale. Neurophysiological experiments have revealed complicated neural machinery dedicated to the computation of motion [1]. In primates, the classical picture of the visual system distinguishes between an object-recognition-focused ventral pathway and an equally large dorsal pathway for object localization and visual motion. In this paper we propose a model for the very first cortical computation in the dorsal pathway: that of direction-selective simple cells in primary visual cortex [2]. We continue a line of models which treats visual motion as a general sequence learning problem and proposes asymmetric Hebbian rules for learning such sequences [3, 4]. We reformulate these earlier models in a generative probabilistic framework which allows us to train them on sequences of natural images. For inference we use an online approximate filtering method which resembles the dynamics of recurrently-connected neural networks.

Previous low-level generative models of image sequences have mostly treated time as a third dimension in a sparse coding problem [5]. These approaches have thus far been difficult to map to neural architecture as they have been implemented with noncausal inference algorithms. Furthermore, the spatiotemporal sensitivity of each learned variable is determined by a separate three-dimensional basis function, requiring many variables to encode all possible orientations, directions of motion and speeds. Cortical architecture points to a more distributed formation of motion representation, with temporal sensitivity determined by the interaction of neurons with different spatial receptive fields. Another major line of generative models of video analyzes the slowly changing features of visual input and proposes complex cells as such slow feature learners [6], [7]. However, these models are not expressive enough to encode visual motion and are more specifically designed to discover image dimensions invariant in time.

A recent hierarchical generative model for mid-level visual motion separates the phases and amplitudes of complex coefficients applied to complex spatial basis functions [8]. This separation makes it possible to build a second layer of variables that specifies a distribution on the phase coefficients alone. This second layer learns to pool together first layer neurons with similar preferred directions. The introduction of real and imaginary parts in the basis functions is inspired by older energy-based approaches where pairs of neurons with receptive fields in quadrature phase feed their outputs with different time delays to a higher-order neuron which thus acquires direction selectivity. The model of [8], and models based on motion energy in general, do not however reproduce direction-selective *simple* cells. In this paper we propose a network in which local motion is computed in a more distributed fashion than is postulated by feedforward implementations of energy models.

## 1.1 Recurrent Network Models for Neural Sequence Learning.

Another view of the development of visual motion processing sees it as a special case of the general problem of sequence learning [4]. Many structures in the brain seem to show various forms of sequence learning, and recurrent networks of neurons can naturally produce learned sequences through their dynamics [9, 10]. Indeed, it has been suggested that the reproduction of remembered sequences within the hippocampus has an important navigational role. Similarly, motor systems must be able to generate sequences of control signals that drive appropriate muscle activity. Thus many neural sequence models are fundamentally generative. By contrast, it is not evident that V1 should need to reproduce the learnt sequences of retinal input that represent visual motion. Although generative modelling provides a powerful mathematical device for the construction of inferential sensory representations, the role of actual generation has been debated. Is there really a potential connection then, to the generative sequence reproduction models developed for other areas?

One possible role for explicit sequence generation in a sensory system is for prediction. Predictive coding has indeed been proposed as a central mechanism to visual processing [11] and even as a more general theory of cortical responses [12]. More specifically as a visual motion learning mechanism, sequence learning forms the basis of an earlier simple toy but biophysically realistic model based on STDP at the lateral synapses of a recurrently connected network [4]. In another biophysically realistic model, recurrent connections are set by hand rather than learned but they produce direction selectivity and speed tuning in simulations of cat primary visual cortex [13]. Thus, the recurrent mechanisms of sequence learning may indeed be important. In the following section we define mathematically a probabilistic sequence modelling network which can learn patterns of visual motion in an unsupervised manner from 16 by 16 patches with 512 latent variables connected densely to each other in a nonlinear dynamical system.

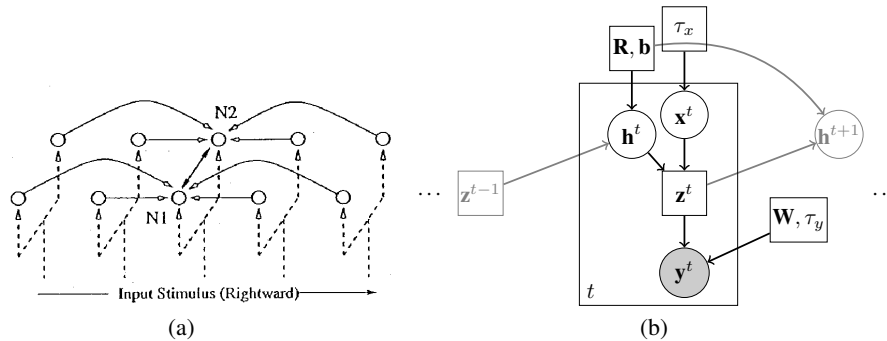

(a)                                              (b)

Figure 1: **a.** Toy sequence learning model with biophysically realistic neurons from [4]. Neurons N1 and N2 have the same RF as indicated by the dotted line, but after STDP learning of the recurrent connections with other neurons in the chain, N1 and N2 learn to fire only for rightward and respectively leftward motion. **b** Graphical model representation of the bgG-RNN. The square box represents that the variable $\mathbf{z}_t$ is not random, but is given by $\mathbf{z}_t = \mathbf{x}_t \circ \mathbf{h}_t$.

## 2 Probabilistic Recurrent Neural Networks

In this section we introduce the binary-gated Gaussian recurrent neural network as a generative model of sequences of images. The model belongs to the class of nonlinear dynamical systems. Inference methods in such models typically require expensive variational [14] or sampling based approximations [15], but we found that a low cost online filtering method works sufficiently well to learn an interesting model. We begin with a description of binary-gated Gaussian sparse coding for still images and then describe how to define the dependencies in time between variables.

### 2.1 Binary Gated Gaussian Sparse Coding (bgG-SC).

Binary-gated Gaussian sparse coding ([16], also called spike-and-slab sparse coding [17][1]) may be seen as a limit of sparse coding with a mixture of Gaussians priors [18] where one mixture component has zero variance. Mathematically, the data $\mathbf{y}^t$ is obtained by multiplying together a matrix $\mathbf{W}$ of basis filters with a vector $\mathbf{h}^t \circ \mathbf{x}^t$, where $\circ$ denotes the operation of Hadamard or element-wise product, $\mathbf{x}^t \in \mathbb{R}^N$ is Gaussian and spherically distributed with standard deviation $\tau_x$ and $\mathbf{h}^t \in \{0,1\}^N$ is a vector of independent Bernoulli-distributed elements with success probabilities $\mathbf{p}$. Finally, small amounts of isotropic Gaussian noise with standard deviation $\tau_y$ are added to produce $\mathbf{y}^t$. For notational consistency with the proposed dynamic version of this model, the $t$ superscript indexes time. The joint log-likelihood is

$$\mathcal{L}_{\text{SC}}^t = - \|\mathbf{y}^t - \mathbf{W} \cdot (\mathbf{h}^t \circ \mathbf{x}^t)\|^2 / 2\tau_y^2 - \|\mathbf{x}^t\|^2 / 2\tau_x^2 +$$
$$+ \sum_{j=1}^{N} \left( h_j^t \log p_j + (1 - h_j^t) \log (1 - p_j) \right) + \text{const}, \tag{1}$$

where $N$ is the number of basis filters in the model. By using appropriately small activation probabilities $\mathbf{p}$, the effective prior on $\mathbf{h}^t \circ \mathbf{x}^t$ can be made arbitrarily sparse. Probabilistic inference in sparse coding is intractable but efficient variational approximation methods exist. We use a very fast approximation to MAP inference, the matching pursuit algorithm (MP) [19]. Instead of using MP to extract a fixed number of coefficients per patch as usual, we extract coefficients for as long as the joint log-likelihood increases. Patches with more complicated structure will naturally require more coefficients to code. Once values for $\mathbf{x}^t$ and $\mathbf{h}^t$ are filled in, the gradient of the joint log likelihood with respect to the parameters is easy to derive. Note that $x_k^t$ for which $h_k^t = 0$ can be integrated out in the likelihood, as they receive no contribution from the data term in (1). Due to the MAP approximation, only the $\mathbf{W}$ can be learned. Therefore, we set $\tau_x^2, \tau_y^2$ to reasonable values, both on the order of the data variance. We also adapted $p_k$ during learning so that each filter was selected by the MP process a roughly equal number of times. This helped to stabilise learning, avoiding a tendency to very unequal convergence rates otherwise encountered.

When applied to whitened small patches from images, the algorithm produced localized Gabor-like receptive fields as usual for sparse coding, with a range of frequencies, phases, widths and aspect ratios. The same model was shown in [16] to reproduce the diverse shapes of V1 receptive fields, unlike standard sparse coding [20]. We found that when we varied the average number of coefficients recruited per image, the receptive fields of the learned filters varied in size. For example with only one coefficient per image, a large number of filters represented edges extending from one end of the patch to the other. With a large number of coefficients, the filters concentrated their mass around just a few pixels. With even more coefficients, the learned filters gradually became Fourier-like.

During learning, we gradually adapted the average activation of each variable $h_k^t$ by changing the prior activation probabilities $p_k$. For 16x16 patches in a twice overcomplete SC model (number of filters = twice the number of pixels), we found that learning with 10-50 coefficients on average prevented the filters from becoming too much or too little localized in space.

## 2.2 Binary-Gated Gaussian Recurrent Neural Network (bgG-RNN).

To obtain a dynamic hidden model for sequences of images $\{\mathbf{y}^t\}$ we specify the following conditional probabilities between hidden chains of variables $\mathbf{h}^t, \mathbf{x}^t$:

$$\mathbf{P}\left(\mathbf{x}^{t+1}, \mathbf{h}^{t+1} | \mathbf{x}^t, \mathbf{h}^t\right) = \mathbf{P}\left(\mathbf{x}^{t+1}\right) \mathbf{P}\left(\mathbf{h}^{t+1} | \mathbf{h}^t \circ \mathbf{x}^t\right)$$
$$\mathbf{P}\left(\mathbf{x}^{t+1}\right) = \mathcal{N}\left(\mathbf{0}, \tau_x^2 \mathbf{I}\right)$$
$$\mathbf{P}\left(\mathbf{h}^{t+1} | \mathbf{h}^t \circ \mathbf{x}^t\right) = \sigma\left(\mathbf{R} \cdot \left(\mathbf{h}^t \circ \mathbf{x}^t\right) + \mathbf{b}\right), \tag{2}$$

where $\mathbf{R}$ is a matrix of recurrent connections, $\mathbf{b}$ is a vector of biases and $\sigma$ is the standard sigmoid function $\sigma(a) = 1/(1 + \exp(-a))$. Note how the $\mathbf{x}^t$ are always drawn independently while the conditional probability for $\mathbf{h}^{t+1}$ depends only on $\mathbf{h}^t \circ \mathbf{x}^t$. We arrived at these designs based on a few observations. First, similar to inference in bgG-RNN, the conditional dependence on $\mathbf{h}^t \circ \mathbf{x}^t$, allows us to integrate out variables $\mathbf{x}^t, \mathbf{x}^{t+1}$ for which the respective gates in $\mathbf{h}^t, \mathbf{h}^{t+1}$ are 0. Second, we observed that adding Gaussian linear dependencies between $\mathbf{x}^{t+1}$ and $\mathbf{x}^t \circ \mathbf{h}^t$ did not modify qualitatively the results reported here. However, dropping $\mathbf{P}\left(\mathbf{h}^{t+1} | \mathbf{h}^t \circ \mathbf{x}^t\right)$ in favor of $\mathbf{P}\left(\mathbf{x}^{t+1} | \mathbf{h}^t \circ \mathbf{x}^t\right)$ resulted in a model which could no longer learn a direction-selective representation. For simplicity we chose the minimal model specified by (2). The full log likelihood for the bgG-RNN is $\mathcal{L}_{\text{bgG-RNN}} = \sum_t \mathcal{L}_{\text{bgG-RNN}}^t$ where

$$\mathcal{L}_{\text{bgG-RNN}}^t = \text{const} - \|\mathbf{y}^t - \mathbf{W}(\mathbf{x}^t \circ \mathbf{h}^t)\|^2 / 2\tau_y^2 - \|\mathbf{x}^t\|^2 / 2\tau_x^2$$
$$+ \sum_{j=1}^{N} h_j^t \log \sigma \left(\mathbf{R}\left(\mathbf{h}^{t-1} \circ \mathbf{x}^{t-1}\right) + \mathbf{b}\right)_j$$
$$+ \sum_{j=1}^{N} (1 - h_j^t) \log \left(1 - \sigma\left(\mathbf{R}\left(\mathbf{h}^{t-1} \circ \mathbf{x}^{t-1}\right) + \mathbf{b}\right)_j\right), \tag{3}$$

where $\mathbf{x}^0 = \mathbf{0}$ and $\mathbf{h}^0 = \mathbf{0}$ are both defined to be vectors of zeros.

## 2.3 Inference and learning of bgG-RNN.

The goal of inference is to set the values of $\hat{\mathbf{x}}^t, \hat{\mathbf{h}}^t$ for all $t$ in such a way as to minimize the objective set by (3). Assuming we have already set $\hat{\mathbf{x}}^t, \hat{\mathbf{h}}^t$ for $t = 1$ to $T$, we propose to obtain $\hat{\mathbf{x}}^{T+1}, \hat{\mathbf{h}}^{T+1}$ exclusively from $\hat{\mathbf{x}}^T, \hat{\mathbf{h}}^T$. This scheme might be called greedy filtering. In greedy filtering, inference is causal and Markov with respect to time. At step $T + 1$ we only need to solve a simple SC problem given by the slice $\mathcal{L}_{\text{bgG-RNN}}^{T+1}$ of the likelihood (3), where $\mathbf{x}^T, \mathbf{h}^T$ have been replaced with the estimates $\hat{\mathbf{x}}^T, \hat{\mathbf{h}}^T$. The greedy filtering algorithm proposed here scales linearly with the number of time steps considered and is well suited for online inference. The algorithm might not produce very accurate estimates of the global MAP settings of the hidden variables, but we found it was sufficient for learning a complex bgG-RNN model. In addition, its simplicity coupled with the fast MP algorithm in each $\mathcal{L}_{\text{bgG-RNN}}^t$ slice, resulted in very fast inference and consequently fast learning. We usually learned models in under one hour on a quad core workstation.

Due to our approximate inference scheme, some parameters in the model had to be set manually. These are $\tau_x^2$ and $\tau_y^2$, which control the relative strengths in the likelihood of three terms: the data likelihood, the smallness prior on the Gaussian variables and the interaction between sets of $\mathbf{x}^t, \mathbf{h}^t$ consecutive in time. In our experiments we set $\tau_y^2$ equal to the data variance and $\tau_x^2 = 2\tau_y^2$. We found that such large levels of expected observation noise were necessary to drive robust learning in $\mathbf{R}$.

For learning we initialized parameters randomly to small values and first learned $\mathbf{W}$ exclusively. Once the filters converge, we turn on learning for $\mathbf{R}$. $\mathbf{W}$ does not change very much beyond this point. We found learning of $\mathbf{R}$ was sensitive to learning rate. We set the learning rate to $0.05$ per batch, used a momentum term of $0.75$ and batches of 30 sets of 100 frame sequences. We stabilized the mean activation probability of each neuron individually by actively and quickly tuning the biases

**b** during learning. We whitened images with a center-surround filter and standardized the whitened pixel values.

Gradients required for learning **R** show similarities to the STDP learning rule used in [3] and [4].

$$\frac{\partial \mathcal{L}^t_{\text{bgG-RNN}}}{\partial R_{jk}} = \left( h_k^{t-1} x_k^{t-1} \right) \cdot \left( h_j^t - \sigma \left( \mathbf{R} \left( \mathbf{h}^t \circ \mathbf{x}^t \right) + \mathbf{b} \right)_j \right). \tag{4}$$

We will assume for neural interpretation that the positive and negative values of $\mathbf{x}^t \circ \mathbf{h}^t$ are encoded by different neurons. If for a given neuron $x_k^{t-1}$ is always positive, then the gradient (4) is only strictly positive when $h_k^{t-1} = 1$ and $h_j^t = 1$ and strictly negative when $h_k^{t-1} = 1$ and $h_j^t = 0$. In other words, the connection $R_{jk}$ is strengthened when neuron $k$ appears to cause neuron $j$ to activate and inhibited if neuron $k$ fails to activate neuron $j$. A similar effect can be observed for the negative part of $x_k^{t-1}$. This kind of Hebbian rule is widespread in cortex for long term learning and is used in previous computational models of neural sequence learning that partly motivated our work [4].

## 3    Results

For data, we selected about 100 short 100 frame long clips from a high resolution BBC wild life documentary. Clips were chosen only if they seemed on visual inspection to have sufficient motion energy over the 100 frames. The clips chosen ended up being mostly panning shots and close-ups of animals in their natural habitats (the closer the camera is to a moving object, the faster it appears to move).

The results presented below measure the ability of the model to produce responses similar to those of neurons recorded in primate experiments. The stimuli used in these experiments are typically of two kinds: drifting gratings presented inside circular or square apertures or translating bars of various lengths. These two kinds of stimuli produce very clear motion signals, unlike motion produced by natural movies. In fact, most patches we used in training contained a wide range of spatial orientations, most of which were not orthogonal to the direction of local translation. After comparing model responses to neural data, we finish with an analysis of the network connectivity pattern that underlies the responses of model neurons.

### 3.1    Measuring responses in the model.

We needed to deal with the potential negativity of the variables in the model, since neural responses are always positive quantities. We decided to separate the positive and negative parts of the Gaussian variables into two distinct sets of responses. This interpretation is relatively common for sparse coding models and we also found that in many units direction selectivity was enhanced when the positive and negative parts of $\mathbf{x}^t$ were separated (as opposed to taking $\mathbf{h}^t$ as the neural response). The enhancement was supported by a particular pattern of network connectivity which we describe in a later subsection.

Since our inference procedure is deterministic it will produce the exact same response to the same stimulus every time. We added Gaussian noise to the spatially whitened test image sequences, partly to capture the noisy environments in cortex and partly to show robustness of direction selectivity to noise. The amount of noise added was about half the expected variance of the stimulus.

### 3.2    Direction selectivity and speed tuning.

Direction selectivity is measured with the following index: $\text{DI} = 1 - R_{\text{opp}}/R_{\text{max}}$. Here $R_{\text{max}}$ represents the response of a neuron in its preferred direction, while $R_{\text{opp}}$ is the response in the direction opposite to that preferred. This selectivity index is commonly used to characterize neural data. To define a neuron's preferred direction, we inferred latent coefficients over many repetitions of square gratings drifting in 24 directions, at speeds ranging from 0 to 3 pixels/frame in 0.25 steps. The periodicity of the stimulus was twice the patch size, so that motion locally appeared as an advancing long edge. The neuron's preferred direction was defined as the direction in which it responded most strongly, averaged over all speeds. Once a preferred direction was established, we defined the neuron's preferred speed, as the speed at which it responded most strongly in its preferred

direction. Finally, at this preferred speed and direction, we calculated the DI of the neuron. Similar results were obtained if we averaged over all speed conditions.

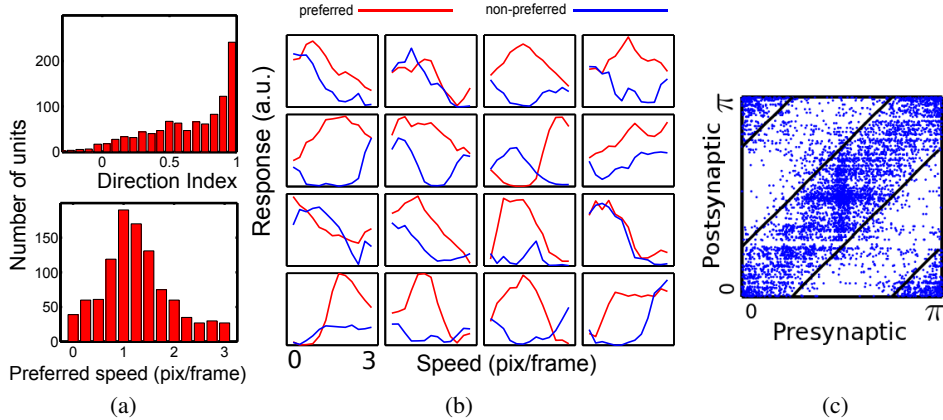

Figure 2: **a.** Speed tuning of 16 randomly chosen neurons. Note that some neurons only respond weakly without motion, some are inhibited in the non-preferred direction compared to static responses and most have a clear peak in the preferred direction at specific speeds. **b.** top: Histogram of direction selectivity indices. bottom: Histogram of preferred speeds. **c.** For each of the 10 strongest excitatory connections per neuron we plot a dot indicating the orientation selectivity of pre and post-synaptic units. Note that most of the points are within $\pi/4$ from the diagonal, an area marked by the black lines. Notice also the relatively increased frequency of horizontal and vertical edges.

We found that most neurons in the model had sharp tuning curves and direction-selective responses. We cross validated the value of the direction index with a new set of responses (fixing the preferred direction) to obtain an averaged DI of 0.65, with many neurons having a DI close to 1 (see figure 2(a)). This distribution is similar to those shown in [21] for real V1 neurons. 714 of 1024 neurons were classified as direction-selective, on the basis of having DI > 0.5. Distributions of direction indices and optimal speeds are shown in figure 2(a). A neuron's preferred direction was always close to orthogonal to the axis of its Gabor receptive field, except for a few degenerate cases around the edges of the patch. We defined the population tuning curve as the average of the tuning curves of individual neurons, each aligned by their preferred direction of motion. The DI of the population was 0.66. Neurons were also speed tuned, in that responses could vary greatly and systematically as a function of speed and DI was non-constant as a function of speed (see figure 2(b)). Usually at low and high speeds the DI was 0, but in between a variety of responses were observed. Speed tuning is also present in recorded V1 neurons [22], and could form the basis for global motion computation based on the intersection of constraints method [23].

## 3.3 Vector velocity tuning.

To get a more detailed description of single-neuron tuning, we investigated responses to different stimulus velocities. Since drifting gratings only contain motion orthogonal to their orientation, we switched to small (1.25pix x 2pix) drifting Gabors for these experiments. We tested the network's behavior with a full set of 24 Gabor orientations, drifting in a full set of 24 directions with speeds ranging from 0.25 pixels/frame to 3 pixels/frame, for a total of 6912 = 24 x 24 x 12 conditions with hundreds of repetitions of each condition. For each neuron we isolated its responses to drifting Gabors of the same orientation travelling at the 12 different speeds in the 24 different directions. We present these for several neurons in polar plots in figure 3(b). Note responses tend to be high to vector velocities lying on a particular line.

## 3.4 Connectomics in silico

We had anticipated that the network would learn direction selectivity via specific patterns of recurrent connection, in a fashion similar to the toy model studied in [4]. We now show that the pattern of connectivity indeed supports this computation.

The most obvious connectivity pattern, clearly visible for single neurons in figure 3(a), shows that neurons in the model excite other neurons in their preferred direction and inhibit neurons in the opposite direction. This asymmetric wiring naturally supports direction selectivity.

Asymmetry is not sufficient for direction selectivity to emerge. In addition, strong excitatory projections have to connect together neurons with similar preferred orientations and similar preferred directions. Only then will direction information propagate in the network in the identities of the active variables (and the signs of their respective coefficients $\mathbf{x}^t$). We considered for each neuron its 10 strongest excitatory *outputs* and calculated the expected deviation between the orientation of these outputs and the orientation of the root neuron. The average deviation was $23°$, half the expected deviation if connections were random. Figure 2(c) shows a raster plot of the pairs of orientations. The same pattern held when we considered the strongest excitatory *inputs* to a given neuron with an expected deviation of orientations of $24°$. We could not directly measure if neurons connected together according to direction selectivity because of the sign ambiguity of $\mathbf{x}^t$ variables. One can visually assess in figure 3(a) that neurons connected asymmetrically with respect to their RF axis, but did they also respond to motion primarily in that direction? As can be seen in figure 3(b), which shows the same neurons as figure 3(a), they did indeed. Direction tuning is a measure of the *incoming* connections to the neuron, while figure 3(a) shows the *outgoing* connections. We can qualitatively assess recurrence primarily connected together neurons with similar direction preferences.

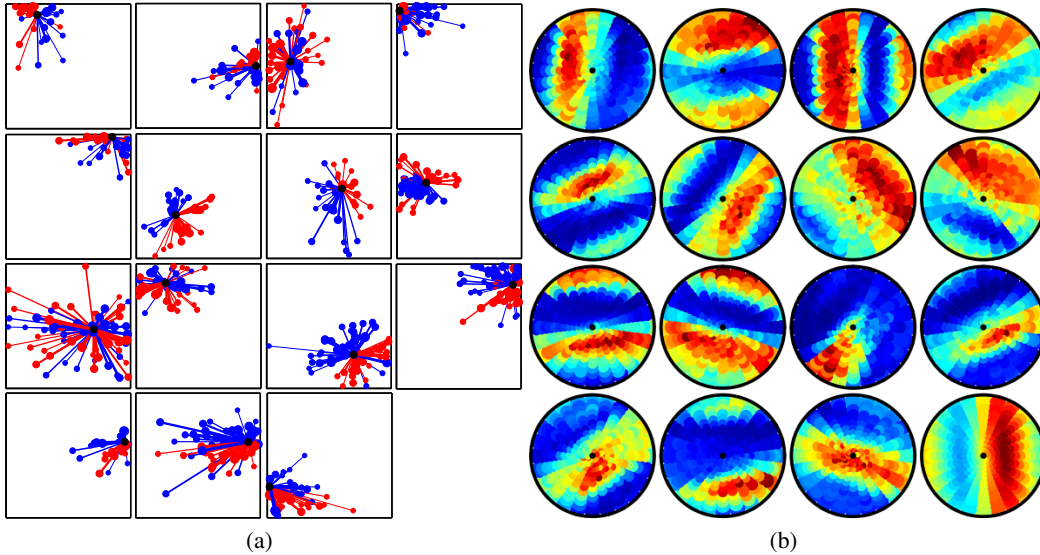

(a)                                                          (b)

Figure 3: **a.** Each plot is based on the outgoing connections of a random set of direction-selective neurons. The centers of the Gabor fits to the neurons' receptive fields are shown as circles on a square representing the 16 by 16 image patch. The root neurons are shown as filled black circles. Filled red/blue circles show neurons to which the root neurons have strong positive/negative connections, with a cutoff at one fourth of the maximal absolute connection. The width of the connecting lines and the area of the filled circles are proportional to the strength of the connection. A dynamic version of this plot during learning is shown as a movie in the supplementary material. **b.** The polar plots show the responses of neurons presented in *a* to small, drifting Gabors that match their respective orientations. Neurons are aligned in exactly the same manner on the 4 by 4 grid. Every small disc in every polar plot represents one combination of speed and direction and the color of the disc represents the magnitude of the response, with intense red being maximal and dark blue minimal. The vector from the center of the polar plot to the center of each disc is proportional to the vector displacement of each consecutive frame in the stimulus sequence. Increasing disc sizes at faster speeds are used for display purposes. The very last polar plot shows the average of the responses of the entire population, when all neurons are aligned by their preferred direction.

We also observed that neurons mostly projected strong excitatory outputs to other neurons that were aligned parallel to the root neuron's main axis (visible in figures 3(a)). We think this is related to the fact that locally all edges appear to translate parallel to themselves. A neuron $X$ with a preferred direction $v$ and preferred speed $s$ has a so-called constraint line (CL), parallel to the Gabor's axis.

When the neuron is activated by an edge $E$, the constraint line is formed by all possible future locations of edge $E$ that are consistent with global motion in the direction $v$ with speed $s$. Due to the presence of long contours in natural scenes, the activation of $X$ can predict at the next time step the activations of other neurons with RFs aligned on the CL. Our likelihood function encourages the model to learn to make such predictions as well as it can. To quantify the degree to which connections were made along a CL, for each neuron we fit a 2D Gaussian to the distribution of RF positions of the 20 most strongly connected neurons (the filled red circles in figure 3(a)), each further weighted by its strength. The major axis of the Gaussians represent the constraint lines of the root neuron and are in 862 out of 1024 neurons less than $15°$ away from perfectly parallel to the root neurons' axis. The distance of each neuron to their constraint line was on average 1.68 pixels.

Yet perhaps the strongest manifestation of the CL tuning property of neurons in the model can be seen in their responses to small stimuli drifting with different vector velocities. Many of the neurons in figure 3(b) respond best when the velocity vector ends on the constraint line and a similar trend holds for the aligned population average.

It is already known from experiments of axon mappings simultaneous with dye-sensitive imaging that neurons in V1 are more likely to connect with neurons of similar orientations situated as far away as 4 mm / 4-8 minicolumns away [24]. The model presented here makes three further predictions: that neurons connect more strongly to neurons in their preferred direction, that connected neurons lie on the constraint line and that they have similar preferred directions to the root neuron.

## 4    Discussion

We have shown that a network of recurrently-connected neurons can learn to discriminate motion direction at the level of individual neurons. Online greedy filtering in the model is a sufficient approximate-inference method to produce direction-selective responses. Fast, causal and online inference is a necessary requirement for practical vision systems (such as the brain) but previous visual-motion models did not provide such an implementation of their inference algorithms. Another shortcoming of these previous models is that they obtain direction selectivity by having variables with different RFs at different time lags, effectively treating time as a third spatial dimension. A dynamic generative model may be more suited for online inference with methods such as particle filtering, assumed density filtering, or the far cheaper method employed here of greedy filtering.

The model neurons can be interpreted as predicting the motion of the stimulus. The lateral inputs they receive are however not sufficient in themselves to produce a response, the prediction also has to be consistent with the bottom-up input. When the two sources of information disagree, the network compromises but not always in favor of the bottom-up input, as this source of information might be noisy. This is reflected by the decrease in reconstruction accuracy from 80% to 60 % after learning the recurrent connections. It is tempting to think of V1 direction selective neurons as not only edge detectors and contour predictors (through the nonclassical RF) but also predictors of future edge locations, through their specific patterns of connectivity.

The source of direction selectivity in cortex is still an unresolved question, but note that in the retina of non-primate mammals it is known with some certainty that recurrent inhibition in the non preferred direction is largely responsible for the direction selectivity of retinal ganglion cells [25]. It is also known that unlike orientation and ocular dominance, direction selectivity requires visual experience to develop [26], perhaps because direction selectivity depends on a specific pattern of lateral connectivity unlike the largely feedforward orientation and binocular tuning. Another experiment showed that after many exposures to the same moving stimulus, the sequence of spikes triggered in different neurons along the motion trajectory was also triggered in the complete absence of motion, again indicating that motion signals in cortex may be generated internally from lateral connections [27].

Thus, we see a number of reasons to propose that direction selectivity in the cortex may indeed develop and be computed through a mechanism analagous to the one we have developed here. If so, then experimental tests of the various predictions developed above should prove to be revealing.

## Footnotes

[1] Although less evocative, we prefer the term 'binary-gated Gaussian' to 'spike-and-slab' partly because our slab is really more of a hump, and partly because the spike refers to a feature seen only in the density of the product $h_i x_i$, rather than in either the values or distributions of the component variables.

# References

[1] A Mikami, WT Newsome, and RH Wurtz. Motion selectivity in macaque visual cortex. II. Spatiotemporal range of directional interactions in MT and V1. *Journal of Neurophysiology*, 55(6):1328–1339, 1986.

[2] MS Livingstone. Mechanisms of direction selectivity in macaque V1. *Neuron*, 20:509–526, 1998.

[3] LF Abbott and KI Blum. Functional significance of long-term potentiation for sequence learning and prediction. *Cerebral Cortex*, 6:406–416, 1996.

[4] RPN Rao and TJ Sejnowski. Predictive sequence learning in recurrent neocortical circuits. *Advances in Neural Information Processing*, 12:164–170, 2000.

[5] B Olshausen. Learning sparse, overcomplete representations of time-varying natural images. *IEEE International Conference on Image Processing*, 2003.

[6] P Berkes, RE Turner, and M Sahani. A structured model of video produces primary visual cortical organisation. *PLoS Computational Biology*, 5, 2009.

[7] L Wiskott and TJ Sejnowski. Slow feature analysis: Unsupervised learning of invariances. *Neural Computation*, 14(4):715–770, 2002.

[8] C Cadieu and B Olshausen. Learning transformational invariants from natural movies. *Advances in Neural Information Processing*, 21:209–216, 2009.

[9] D Barber. Learning in spiking neural assemblies. *Advances in Neural Information Processing*, 15, 2002.

[10] J Brea, W Senn, and JP Pfister. Sequence learning with hidden units in spiking neural networks. *Advances in Neural Information Processing*, 24, 2011.

[11] RP Rao and Ballard DH. Predictive coding in the visual cortex: a functional interpretation of some extra-classical receptive-field effects. *Nature Neuroscience*, 2(1):79–87, 1999.

[12] K Friston. A theory of cortical responses. *Phil. Trans. R. Soc. B*, 360(1456):815–836, 1999.

[13] RJ Douglas, C Koch, M Mahowald, KA Martin, and HH Suarez. Recurrent excitation in neocortical circuits. *Science*, 269(5226):981–985, 1995.

[14] TP Minka. Expectation propagation for approximate Bayesian inference. *UAI'01 Proceedings of the Seventeenth conference on Uncertainty in artificial intelligence*, pages 362–369, 2001.

[15] A Doucet, N Freitas, K Murphy, and S Russell. Rao-blackwellised particle filtering for dynamic Bayesian networks. *UAI'00 Proceedings of the Sixteenth conference on Uncertainty in artificial intelligence*, pages 176–183, 2000.

[16] M Rehn and FT Sommer. A network that uses few active neurones to code visual input predicts the diverse shapes of cortical receptive fields. *Journal of Computational Neuroscience*, 22:135–146, 2007.

[17] IJ Goodfellow, A Courville, and Y Bengio. Spike-and-slab sparse coding for unsupervised feature discovery. arXiv:1201.3382v2, 2012.

[18] BA Olshausen and KJ Millman. Learning sparse codes with a mixture-of-Gaussians prior. *Advances in Neural Information Processing*, 12, 2000.

[19] SG Mallat and Z Zhang. Matching pursuits with time-frequency dictionaries. *IEEE Transactions on Signal Processing*, 41(12):3397–3415, 1993.

[20] BA Olshausen and DJ Field. Emergence of simple-cell receptive field properties by learning a sparse code for natural images. *Nature*, 381:607–609, 1996.

[21] MR Peterson, B Li, and RD Freeman. The derivation of direction selectivity in the striate cortex. *Journal of Neuroscience*, 24:35833591, 2004.

[22] GA Orban, H Kennedy, and J Bullier. Velocity sensitivity and direction selectivity of neurons in areas V1 and V2 of the monkey: influence of eccentricity. *Journal of Neurophysiology*, 56(2):462–480, 1986.

[23] EP Simoncelli and DJ Heeger. A model of neuronal responses in visual area MT. *Vision Research*, 38(5):743–761, 1998.

[24] WH Bosking, Y Zhang, B Schofield, and D FitzPatrick. Orientation selectivity and the arrangement of horizontal connections in tree shrew striate cortex. *Journal of Neuroscience*, 17(6):2112–2127, 1997.

[25] SI Fried, TA Munch, and FS Werblin. Directional selectivity is formed at multiple levels by laterally offset inhibition in the rabbit retina. *Neuron*, 46(1):117–127, 2005.

[26] Y Li, D FitzPatrick, and LE White. The development of direction selectivity in ferret visual cortex requires early visual experience. *Nature Neuroscience*, 9(5):676–681, 2006.

[27] S Xu, W Jiang, M Poo, and Y Dan. Activity recall in a visual cortical ensemble. *Nature Neuroscience*, 15:449–455, 2012.

